# NETWORK MODEL OF STATE-DEPENDENT SEQUENCING

Jeffrey P. Sutton,* Adam N. Mamelak† and  J. Allan Hobson
Laboratory of Neurophysiology and Department of Psychiatry
Harvard Medical School
74 Fenwood Road, Boston, MA 02115

## Abstract

A network model with temporal sequencing and state-dependent modulatory features is described. The model is motivated by neurocognitive data characterizing different states of waking and sleeping. Computer studies demonstrate how unique states of sequencing can exist within the same network under different aminergic and cholinergic modulatory influences. Relationships between state-dependent modulation, memory, sequencing and learning are discussed.

## 1  INTRODUCTION

Models of biological information processing often assume only one mode or state of operation. In general, this state depends upon a high degree of fidelity or modulation among the neural elements. In contrast, real neural networks often have a repertoire of processing states that is greatly affected by the relative balances of various neuromodulators (Selverston, 1988; Harris-Warrick and Marder, 1991). One area where changes in neuromodulation and network behavior are tightly and dramatically coupled is in the sleep-wake cycle (Hobson and Steriade, 1986; Mamelak and Hobson, 1989). This cycle consists of three main states: wake, non-rapid eye

movement (NREM) sleep and rapid eye movement (REM) sleep. Each state is characterized by a unique balance of monoaminergic and cholinergic neuromodulation (Hobson and Steriade, 1986; figure 1). In humans, each state also has characteristic cognitive sequencing properties (Foulkes, 1985; Hobson, 1988; figure 1). An integration and better understanding of the complex relationships between neuromodulation and information sequencing are desirable from both a computational and a neurophysiological perspective. In this paper, we present an initial approach to this difficult neurocognitive problem using a network model.

| STATE | MODULATION | | SEQUENCING |
|---|---|---|---|
| | tonic aminergic ($\beta$) | phasic cholinergic ($\delta$) | |
| WAKE | high | low | progressive<br>A1 $\longrightarrow$ A2 $\dashrightarrow$ A3<br>$\downarrow$ $\leftarrow$ input<br>B1 $\longrightarrow$ B2 |
| NREM SLEEP | intermediate | low | perseverative<br>A1<br>$\nearrow$ $\searrow$<br>A3 $\longleftarrow$ A2 |
| REM SLEEP | low | high | bizarre<br>A1 $\longrightarrow$ A2<br>$\downarrow$ $\leftarrow$ PGO<br>A2/B1<br>PGO $\rightarrow$ $\downarrow$<br>B2 $\longrightarrow$ B3 |

Figure 1: Overview of the three state model which attempts to integrate aspects of neuromodulation and cognitive sequencing. The aminergic and cholinergic systems are important neuromodulators that filter and amplify, as opposed to initiating or carrying, distributed information embedded as memories (eg. $A1$, $A2$, $A3$) in neural networks. In the wake state, a relative aminergic dominance exists and the associated network sequencing is logical and progressive. For example, the sequence $A1 \rightarrow A2$ transitions to $B1 \rightarrow B2$ when an appropriate input (eg. $B1$) is present at a certain time. The NREM state is characterized by an intermediate aminergic-to-cholinergic ratio correlated with ruminative and perseverative sequences. Unexpected or "bizarre" sequences are found in the REM state, wherein phasic cholinergic inputs dominate and are prominent in the ponto-geniculo-occipital (PGO) brain areas. Bizarreness is manifest by incongruous or mixed memories, such as $A2/B1$, and sequence discontinuities, such as $A2 \rightarrow A2/B1 \rightarrow B2$, which may be associated with PGO bursting in the absence of other external input.

## 2   AMINERGIC AND CHOLINERGIC NEUROMODULATION

As outlined in figure 1, there are unique correlations among the aminergic and cholinergic systems and the forms of information sequencing that exist in the states of waking and NREM and REM sleep. The following brief discussion, which undoubtably oversimplifies the complicated and widespread actions of these systems, highlights some basic and relevant principles. Interested readers are referred to the review by Hobson and Steriade (1986) and the article by Hobson *et al.* in this volume for a more detailed presentation.

The biogenic amines, including norepinephrine, serotonin and dopamine, have been implicated as tonic regulators of the signal-to-noise ratio in neural networks (*eg.* Mamelak and Hobson, 1989). Increasing (decreasing) the amount of aminergic modulation improves (worsens) network fidelity (figure 2a). A standard means of modeling this property is by a stochastic or gain factor, analogous to the well-known Boltzmann factor $\beta = 1/kT$, which is present in the network updating rule.

Complex neuromodulatory effects of acetylcholine depend upon the location and types of receptors and channels present in different neurons. One main effect is facilitatory excitation (figure 2b). Mamelak and Hobson (1989) have suggested how the phasic release of acetylcholine, involving the bursting of PGO cells in the brainstem, coupled with tonic aminergic demodulation, could induce bifurcations in information sequencing at the network level. The model described in the next section sets out to test this notion.

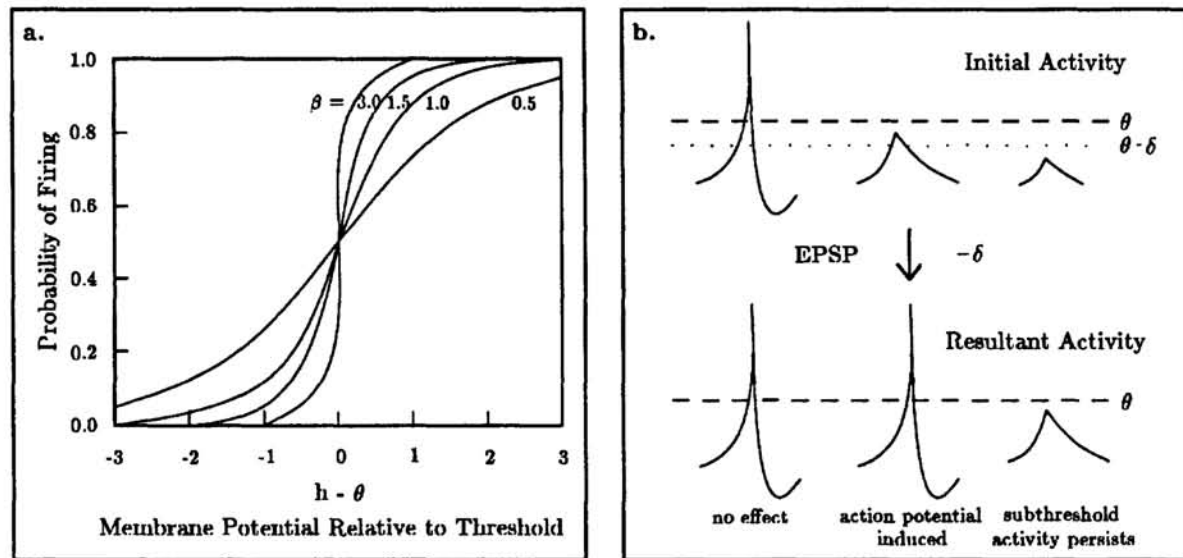

Figure 2: (a) Plot of neural firing probability as a function of the membrane protential, $h$, relative to threshold, $\theta$, for values of aminergic modulation $\beta$ of 0.5, 1.0, 1.5 and 3.0. (b) Schematic diagram of cholinergic facilitation, where EPSPs of magnitude $\delta$ only induce a change in firing activity if $h$ is initially in the range $(\theta - \delta, \theta)$. Modified from Mamelak and Hobson (1989).

## 3   ASSOCIATIVE SEQUENCING NETWORK

There are several ways to approach the problem of modeling modulatory effects on temporal sequencing. We have chosen to commence with an associative network that is an extension of the work on models resembling elementary motor pattern generators (Kleinfeld, 1986; Sompolinsky and Kanter, 1986; Gutfreund and Mezard, 1988). We consider it to be significant that recent data on brainstem control systems show an overlap between sleep-wake regulators and locomotor pattern generators (Garcia-Rill *et al.*, 1990).

The network consists of $N$ neural elements with binary values $S_i = \pm 1$, $i = 1, ..., N$, corresponding to whether they are firing or not firing. The elements are linked together by two kinds of *a priori* learned synaptic connections. One kind,

$$J_{ij}^{(1)} = \frac{1}{N} \sum_{\mu=1}^{p} \xi_i^\mu \xi_j^\mu, \qquad i \neq j, \tag{1}$$

encodes a set of $p$ uncorrelated patterns $\{\xi_i^\mu\}_{i=1}^N$, $\mu = 1, ..., p$, where each $\xi_i^\mu$ takes the value $\pm 1$ with equal probabilities. These patterns correspond to memories that are stable until a transition to another memory is made. Transitions in a sequence of memories $\mu = 1 \rightarrow 2 \rightarrow \cdots \rightarrow q < p$ are induced by a second type of connection

$$J_{ij}^{(2)} = \frac{\lambda}{N} \sum_{\mu=1}^{q-1} \xi_i^{\mu+1} \xi_j^\mu. \tag{2}$$

Here, $\lambda$ is a relative weight of the connection types. The average time spent in a memory pattern before transitioning to the next one in a sequence is $\tau$. At time $t$, the membrane potential is given by

$$h_i(t) = \sum_{j=1}^{N} \left[ J_{ij}^{(1)} S_j(t) + J_{ij}^{(2)} S_j(t - \tau) \right] + \delta_i(t) + I_i(t). \tag{3}$$

The two terms contained in the brackets reflect intrinsic network interactions, while phasic PGO effects are represented by the $\delta_i(t)$. External inputs, other than PGO inputs, to $h_i(t)$ are denoted by $I_i(t)$. Dynamic evolution of the network follows the updating rule

$$S_i(t+1) = \pm 1, \qquad \text{with probability} \qquad \left\{ 1 + e^{\mp 2\beta[h_i(t) - \theta_i(t)]} \right\}^{-1}. \tag{4}$$

In this equation, the amount of aminergic-like modulation is parameterized by $\beta$. While updating could be done serially, a parallel dynamic process is chosen here for convenience. In the absence of external and PGO-like inputs, and with $\beta > 1.0$, the dynamics have the effect of generating trajectories on an adiabatically varying hypersurface that molds in time to produce a path from one basin of attraction to another. For $\beta < 1.0$, the network begins to lose this property. Lowering $\beta$ mostly affects neural elements close to threshold, since the decision to change firing activity centers around the threshold value. However, as $\beta$ decreases, fluctuations in the membrane potentials increase and a larger fraction of the neural elements remain, on average, near threshold.

# 4    SIMULATION RESULTS

A network consisting of $N = 50$ neural elements was examined wherein $p = 6$ memory patterns ($A1$, $A2$, $A3$, $B1$, $B2$ and $B3$) were chosen at random ($p/N = 0.12$). These memories were arranged into two loops, **A** and **B**, according to equation (2) such that the cyclic sequences $A1 \rightarrow A2 \rightarrow A3 \rightarrow A1 \rightarrow \cdots$ and $B1 \rightarrow B2 \rightarrow B3 \rightarrow B1 \rightarrow \cdots$ were stored in loops **A** and **B**, respectively. For simplicity, $\delta_i(t) = \delta(t)$ and $\theta_i(t) = 0$, $\forall i$. The transition parameters were set to $\lambda = 2.5$ and $\tau = 8$ for all the simulations to ensure reliable pattern generation under fully modulated conditions (large $\beta$, $\delta = 0$; Somplinsky and Kanter, 1986). Variations in $\beta$, $\delta(t)$ and $I_i(t)$ delineated the individual states that were examined.

In the model wake state, where there was a high degree of aminergic-like modulation (*eg.* $\beta = 2.0$), the network generated loops of sequential memories. Once cued into one of the two loops, the network would remain in that loop until an external input caused a transition into the other loop (figure 3).

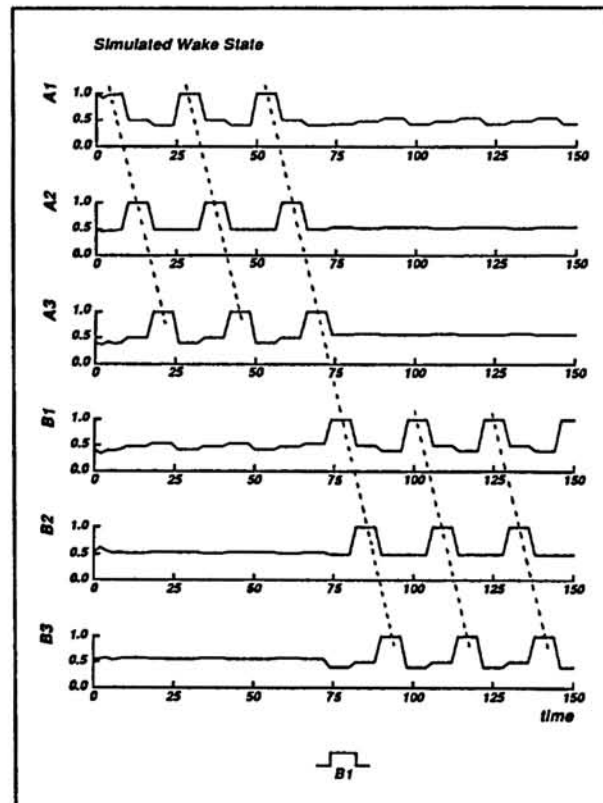

Figure 3: Plot of overlap as a function of time for each of the six memories $A1$, $A2$, $A3$, $B1$, $B2$, $B3$ in the simulated wake state. The overlap is a measure of the normalized Hamming distance between the instantaneous pattern of the network and a given memory. $\beta = 2.0$, $\delta = 0$, $\lambda = 2.5$, $\tau = 8$. The network is cued in pattern $A1$ and then sequences through loop **A**. At $t = 75$, pattern $B1$ is inputted to the network and loop **B** ensues. The dotted lines highlight the transitions between different memory patterns.

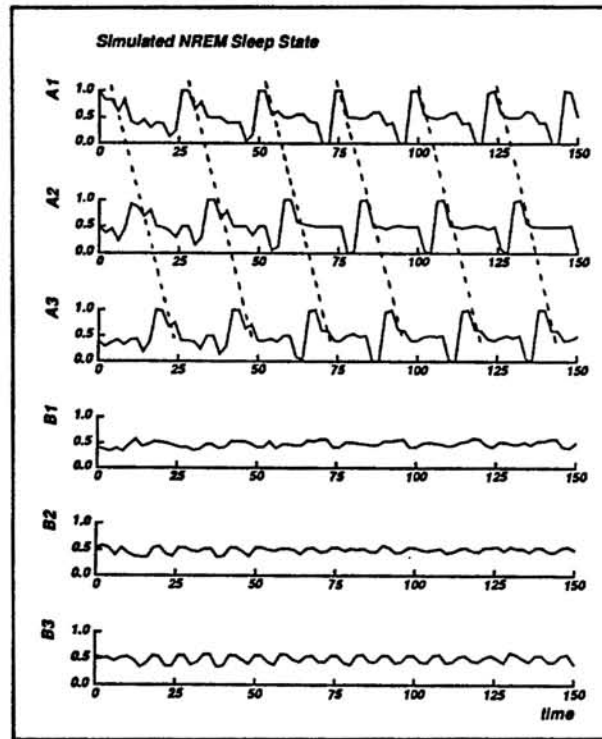

Figure 4: Graph of overlap *vs.* time for each of the six memories in the simulated NREM sleep state. $\beta = 1.1$, $\delta = 0$, $\lambda = 2.5$, $\tau = 8$. Initially, the network is cued in pattern $A1$ and remains in loop **A**. Considerable fluctuations in the overlaps are present and external inputs are absent.

As $\beta$ was decreased (*eg.* $\beta = 1.1$), partially characterizing conditions of a model NREM state, sequencing within a loop was observed to persist (figure 4). However, decreased stability relative to the wake state was observed and small perturbations could cause disruptions within a loop and occasional bifurcations between loops. Nevertheless, in the absence of an effective mechanism to induce inter-loop transitions, the sequences were basically repetitive in this state.

For small $\beta$ (*eg.* $0.8 < \beta < 1.0$) and various PGO-like activities within the simulated REM state, a diverse and rich set of dynamic behaviors was observed, only some of which are reported here. The network was remarkably sensitive to the timing of the PGO type bursts. With $\beta = 1.0$, inputs of $\delta = 2.5$ units in clusters of 20 time steps occurring with a frequency of approximately one cluster per 50 time steps could induce the following: (a) no or little effect on identifiable intra-loop sequencing; (b) bifurcations between loops; (c) a change from orderly intra-loop sequencing to apparent disorder;[1] (d) a change from apparent disorder to orderly progression within a single loop ("defibrillation" effect); (e) a change from a disorderly pattern to another disorderly pattern. An example of transition types (c) and (d), with the overall effect of inducing a bifurcation between the loops, is shown in figure 5.

In general, lower intensity (*eg.* 2.0 to 2.5 units), longer duration (*eg.* >20 time steps) PGO-like bursting was more effective in inducing bifurcations than higher intensity (*eg.* 4.0 units), shorter duration (*eg.* 2 time steps) bursts. PGO induced bifurcations were possible in all states and were associated with significant populations of neural elements that were below, but within $\delta$ units of threshold.

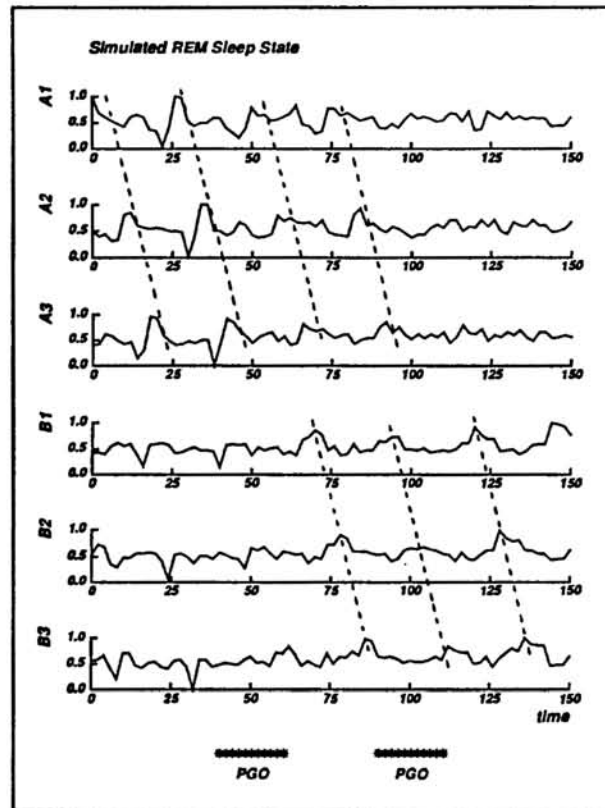

Figure 5: REM sleep state plot of overlap *vs.* time for each of the six memories. $\beta = 1.0$, $\delta = 2.5$, $\lambda = 2.5$, $\tau = 8$. The network sequences progressively in loop **A** until a cluster of simulated PGO bursts (asterisks) occurs lasting $40 < t < 60$. A complex output involving alternating sequences from loop **A** and loop **B** results (note dotted lines). A second PGO burst cluster during the interval $90 < t < 110$ yields an output consisting of a single loop **B** sequence. Over the time span of the simulation, a bifurcation from loop **A** to loop **B** has been induced.

## 5   STATE-DEPENDENT LEARNING

The connections set up by equations (1) and (2) are determined *a priori* using a standard Hebbian learning algorithm and are not altered during the network simulations. Since neuromodulators, including the monoamines norepinephrine and serotonin, have been implicated as essential factors in synaptic plasticity (Kandel *et al.*, 1987), it seems reasonable that state changes in modulation may also affect changes in plasticity. This property, when superimposed on the various sequencing features of a network, may yield possibly novel memory and sequence formations, associations and perhaps other unexamined global processes.

## 6  CONCLUSIONS

The main finding of this paper is that unique states of information sequencing can exist within the same network under different modulatory conditions. This result holds even though the model makes significant simplifying assumptions about the neurophysiological and cognitive processes motivating its construction. Several observations from the model also suggest mechanisms whereby interactions between the aminergic and cholinergic systems can give rise to sequencing properties, such as discontinuities, in different states, especially REM sleep. Finally, the model provides a means of investigating some of the complex and interesting relationships between modulation, memory, sequencing and learning within and between different states.

### Acknowledgements

Supported by NIH grant MH 13,923, the HMS/MMHC Research & Education Fund, the Livingston, Dupont-Warren and McDonnell-Pew Foundations, DARPA under ONR contract N00014-85-K-0124, the Sloan Foundation and Whitaker College.

## Footnotes

*Also in the Center for Biological Information Processing, Whitaker College, E25-201, Massachusetts Institute of Technology, Cambridge, MA 02139

†Currently in the Department of Neurosurgery, University of California, San Francisco, CA 94143

[1] On detailed inspection, the apparent disorder actually revealed several sequences in loops **A** and/or **B** running out of phase with relative delays generally less than $\tau$.

### References

Foulkes D (1985) *Dreaming: A Cognitive-Psychological Analysis.* Hillsdale: Erlbaum.

Garcia-Rill E, Atsuta Y, Iwahara T, Skinner RD (1990) Development of brainstem modulation of locomotion. *Somatosensory Motor Research* 7 238-239.

Gutfreund H, Mezard M (1988) Processing of temporal sequences in neural networks. *Phys Rev Lett* 61 235-238.

Harris-Warrick RM, Marder E (1991) Modulation of neural networks for behavior. *Annu Rev Neurosci* 14 39-57.

Hobson JA (1988) *The Dreaming Brain.* New York: Basic.

Hobson JA, Steriade M (1986) Neuronal basis of behavioral state control. In: Mountcastle VB (ed) *Handbook of Physiology - The Nervous System, Vol IV.* Bethesda: Am Physiol Soc, 701-823.

Kandel ER, Klein M, Hochner B, Shuster M, Siegelbaum S, Hawkins R, *et al.* (1987) Synaptic modulation and learning: New insights into synaptic transmission from the study of behavior. In: Edelman GM, Gall WE, Cowan WM (eds) *Synaptic Function.* New York: Wiley, 471-518.

Kleinfeld D (1986) Sequential state generation by model neural networks. *Proc Natl Acad Sci USA* 83 9469-9473.

Mamelak AN, Hobson JA (1989) Dream bizarrenes as the cognitive correlate of altered neuronal behavior in REM sleep. *J Cog Neurosci* 1(3) 201-22.

Selverston AI (1988) A consideration of invertebrate central pattern generators as computational data bases. *Neural Networks* 1 109-117.

Sompolinsky H, Kanter I (1986) Temporal association in asymmetric neural networks. *Phys Rev Lett* 57 2861-2864.